# Language Induction by Phase Transition in Dynamical Recognizers

Jordan B. Pollack
Laboratory for AI Research
The Ohio State University
Columbus, OH 43210
pollack@cis.ohio-state.edu

## Abstract

A higher order recurrent neural network architecture learns to recognize and generate languages after being "trained" on categorized exemplars. Studying these networks from the perspective of dynamical systems yields two interesting discoveries: First, a longitudinal examination of the learning process illustrates a new form of mechanical inference: Induction by phase transition. A small weight adjustment causes a "bifurcation" in the limit behavior of the network. This phase transition corresponds to the onset of the network's capacity for generalizing to arbitrary-length strings. Second, a study of the automata resulting from the acquisition of previously published languages indicates that while the architecture is NOT guaranteed to find a minimal finite automata consistent with the given exemplars, which is an NP-Hard problem, the architecture does appear capable of generating non-regular languages by exploiting fractal and chaotic dynamics. I end the paper with a hypothesis relating linguistic generative capacity to the behavioral regimes of non-linear dynamical systems.

## 1 Introduction

I expose a recurrent high-order back-propagation network to both positive and negative examples of boolean strings, and report that although the network does **not** find the minimal-description finite state automata for the languages (which is NP-Hard (Angluin, 1978)), it does induction in a novel and interesting fashion, and searches through a hypothesis space which, theoretically, is not constrained to machines of finite state. These results are of import to many related neural models currently under development, e.g. (Elman, 1990; Giles et al., 1990; Servan-Schreiber et al., 1989), and relates ultimately to the question of how linguistic capacity can arise in nature.

Although the transitions among states in a finite-state automata are usually thought of as being fully specified by a table, a transition function can also be specified as a mathematical function of the current state and the input. It is known from (McCulloch & Pitts, 1943) that even the most elementary modeling assumptions yield finite-state

control, and it is worth reiterating that any network with the capacity to compute arbitrary boolean functions (say, as logical sums of products) lapedes farber how nets ], white hornik .], can be used recurrently to implement arbitrary finite state machines.

From a different point of view, a recurrent network with a state evolving across $k$ units can be considered a k-dimensional discrete-time continuous-space dynamical system, with a precise initial condition, $z_k(0)$, and a state space in $Z$, a subspace of $R^k$. The governing function, $F$, is parameterized by a set of weights, $W$, and merely computes the next state from the current state and input, $y_j(t)$, a finite sequence of patterns representing tokens from some alphabet $\Sigma$:

$$z_k(t+1) = F_W(z_k(t), y_j(t))$$

If we view one of the dimensions of this system, say $z_a$, as an "acceptance" dimension, we can define the language accepted by such a *Dynamical Recognizer* as all strings of input tokens evolved from the precise initial state for which the accepting dimension of the state is above a certain threshold. In network terms, one output unit would be subjected to a threshold test after processing a sequence of input patterns.

The first question to ask is how can such a dynamical system be constructed, or taught, to accept a particular language? The weights in the network, individually, do not correspond directly to graph transitions or to phrase structure rules. The second question to ask is what sort of generative power can be achieved by such systems?

## 2    The Model

To begin to answer the question of learning, I now present and elaborate upon my earlier work on Cascaded Networks (Pollack, 1987), which were used in a recurrent fashion to learn parity, depth-limited parenthesis balancing, and to map between word sequences and proposition representations (Pollack, 1990a). A Cascaded Network is a well-controlled higher-order connectionist architecture to which the back-propagation technique of weight adjustment (Rumelhart et al., 1986) can be applied. Basically, it consists of two subnetworks: The *function network* is a standard feed-forward network, with or without hidden layers. However, the weights are dynamically computed by the linear *context network*, whose outputs are mapped in a 1:1 fashion to the weights of the function net. Thus the input pattern to the context network is used to "multiplex" the the function computed, which can result in simpler learning tasks.

When the outputs of the function network are used as inputs to context network, a system can be built which learns to produce specific outputs for variable-length sequences of inputs. Because of the multiplicative connections, each input is, in effect, processed by a different function. Given an initial context, $z_k(0)$, and a sequence of inputs, $y_j(t)$, $t = 1...n$, the network computes a sequence of state vectors, $z_i(t)$, $t = 1...n$ by dynamically changing the set of weights, $w_{ij}(t)$. Without hidden units the forward pass computation is:

$$w_{ij}(t) = \sum_k w_{ijk} z_k(t-1)$$

$$z_i(t) = g(\sum_j w_{ij}(t) y_j(t))$$

where $g$ is the usual sigmoid function used in back-propagation system.

In previous work, I assumed that a teacher could supply a consistent and generalizable desired-state for each member of a large set of strings, which was a significant overconstraint. In learning a two-state machine like parity, this did not matter, as the 1-bit state fully determines the output. However, for the case of a higher-dimensional system, we know what the final output of a system should be, but we *don't care* what its state should be along the way.

Jordan (1986) showed how recurrent back-propagation networks could be trained with "don't care" conditions. If there is no specific preference for the value of an output unit for a particular training example, simply consider the error term for that unit to be 0. This will work, *as long as that same unit receives feedback from other examples*. When the don't-cares line up, the weights to those units will never change. My solution to this problem involves a *backspace*, unrolling the loop only once: After propagating the errors determined on only a subset of the weights from the "acceptance" unit $z_a$:

$$\frac{\partial E}{\partial z_{aj}(n)} = (z_a(n) - d_a) \, z_a(n) \, (1 - z_a(n)) \, y_j(n)$$

$$\frac{\partial E}{\partial w_{ajk}} = \frac{\partial E}{\partial w_{aj}(n)} \, z_k(n-1)$$

The error on the remainder of the weights $(\frac{\partial E}{\partial w_{ijk}}, \ i \neq a \ )$ is calculated using values from the penultimate time step:

$$\frac{\partial E}{\partial z_k(n-1)} = \sum_a \sum_j \frac{\partial E}{\partial w_{ajk}} \, \frac{\partial E}{\partial w_{aj}(n)}$$

$$\frac{\partial E}{\partial w_{ij}(n-1)} = \frac{\partial E}{\partial z_i(n-1)} \, y_j(n-1)$$

$$\frac{\partial E}{\partial w_{ijk}} = \frac{\partial E}{\partial w_{ij}(n-1)} \, z_k(n-2)$$

This is done, in batch (epoch) style, for a set of examples of varying lengths.

## 3   Induction as Phase Transition

In initial studies of learning the simple regular language of odd parity, I expected the recognizer to merely implement "exclusive or" with a feedback link. It turns out that this is not quite enough. Because termination of back-propagation is usually defined as a 20% error (e.g. logical "1" is above 0.8) recurrent use of this logic tends to a limit point. In other words, mere separation of the exemplars is no guarantee that the network can recognize parity in the limit. Nevertheless, this is indeed possible as illustrated by illustrated below. In order to test the limit behavior of a recognizer, we can observe its response to a very long "characteristic string". For odd parity, the string 1* requires an alternation of responses.

A small cascaded network composed of a 1-2 function net and a 2-6 context net

(requiring 18 weights) was was trained on odd parity of a small set of strings up to length 5. At each epoch, the weights in the network were saved in a file. Subsequently, each configuration was tested in its response to the first 25 characteristic strings. In figure 1, each vertical column, corresponding to an epoch, contains 25 points between 0 and 1. Initially, all strings longer than length 1 are not distinguished. From cycle 60-80, the network is improving at separating finite strings. At cycle 85, the network undergoes a "bifurcation," where the small change in weights of a single epoch leads to a phase transition from a limit point to a limit cycle.[1] This phase transition is so "adaptive" to the classification task that the network rapidly exploits it.

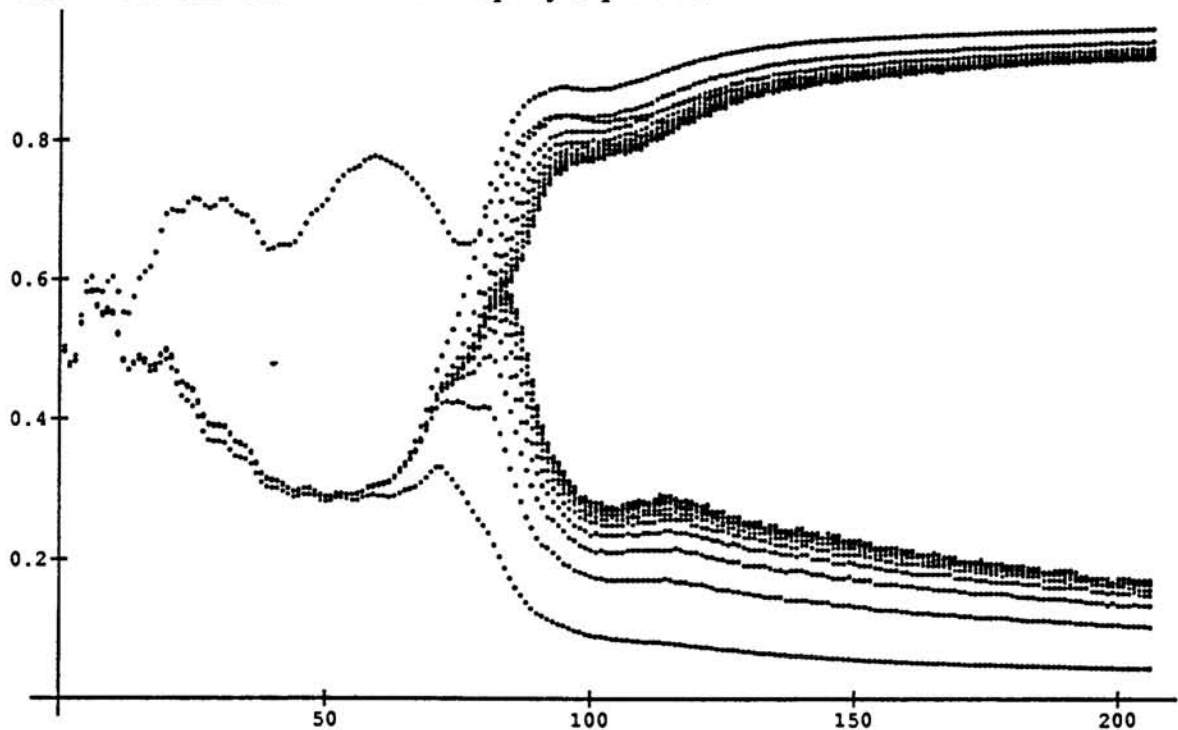

Figure 1:    A bifurcation diagram showing the response of the parity-learner to the first 25 characteristic strings over 200 epochs of training.

I wish to stress that this is a new and very interesting form of mechanical induction, and reveals that with the proper perspective, non-linear connectionist networks are capable of much more complex behavior than hill-climbing. Before the phase transition, the machine is in principle not capable of performing the serial parity task; after the phase transition it **is**. The similarity of learning through a "flash of insight" to biological change through a "punctuated" evolution is much more than coincidence.

## 4   Benchmarking Results

Tomita (1982) performed elegant experiments in inducing finite automata from positive and negative evidence using hillclimbing in the space of 9-state automata. Each case was defined by two sets of boolean strings, accepted by and rejected by the regular languages

listed below.

| 1 | 1* |
|---|---|
| 2 | (1 0)* |
| 3 | no odd zero strings after odd 1 strings |
| 4 | no triples of zeros |
| 5 | pairwise, an even sum of 01's and 10's. |
| 6 | number of 1's - number of 0's = 3n |
| 7 | 0*1*0*1* |

Rather than inventing my own training data, or sampling these languages for a well-formed training set I ran all 7 Tomita training environments as given, on a sequential cascaded network of a 1-input 4-output function network (with bias, 8 weights to set) and a 3-input 8-output context network with bias, using a learning rate was of 0.3 and a momentum to 0.7. Termination was when all accepted strings returned output bits above 0.8 and rejected strings below 0.2.

Of Tomita's 7 cases, all but cases #2 and #6 converged without a problem in several hundred epochs. Case 2 would not converge, and kept treating a negative case as correct because of the difficulty for my architecture to induce a "trap" state; I had to modify the training set (by added reject strings 110 and 11010) in order to overcome this problem.[2] Case 6 took several restarts and thousands of cycles to converge, cause unknown. The complete experimental data is available in a longer report (Pollack, 1990b).

Because the states are "in a box" of low dimension,[3] we can view these machines graphically to gain some understanding of how the state space is being arranged. Based upon some intitial studies of parity, my initial hypothesis was that a set of clusters would be found, organized in some geometric fashion: i.e. an embedding of a finite state machine into a finite dimensional geometry such that each token's transitions would correspond to a simple transformation of space. Graphs of the states visited by all possible inputs up to length 10, for the 7 Tomita test cases are shown in figure 2. Each figure contains 2048 points, but often they overlap.

The images (a) and (d) are what were expected, clumps of points which closely map to states of equivalent FSA's. Images (b) and (e) have limit "ravine's" which can each be considered states as well.

## 5 Discussion

However, the state spaces, (c), (f), and (g) of the dynamical recognizers for Tomita cases 3, 6, and 7, are interesting, because, theoretically, they are *infinite* state machines, where the states are not arbitrary or random, requiring an infinite table of transitions, but are constrained in a powerful way by mathematical principle. In other words, the complexity is in the dynamics, not in the specifications (weights).

In thinking about such a principle, consider other systems in which extreme observed complexity emerges from algorithmic simplicity plus computational power. It is

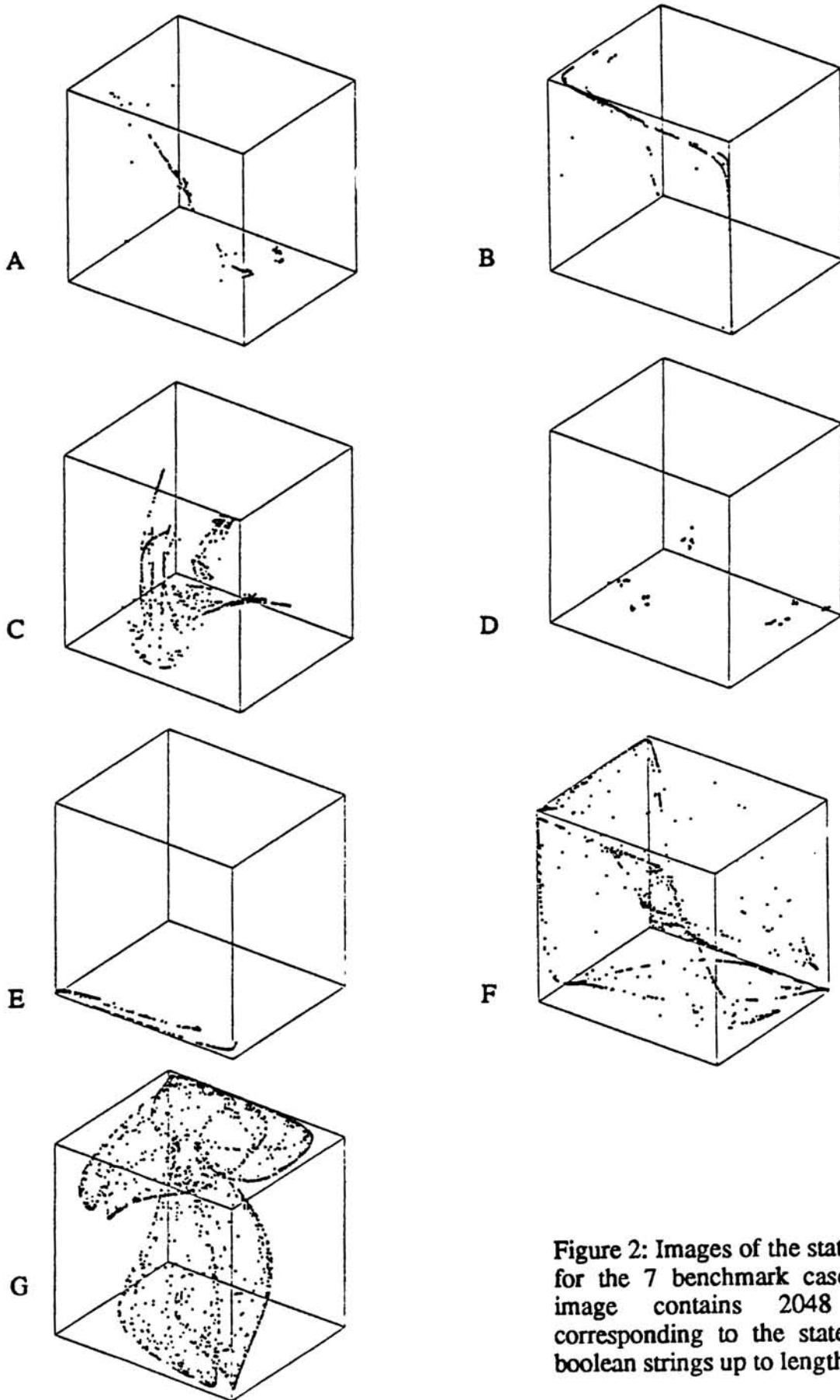

Figure 2: Images of the state-spaces for the 7 benchmark cases. Each image contains 2048 points corresponding to the states of all boolean strings up to length 10.

interesting to note that by eliminating the sigmoid and commuting the $y_j$ and $z_k$ terms, the forward equation for higher order recurrent networks with is identical to the generator of an Iterated Function System (IFS) (Barnsley et al., 1985). Thus, my figures of state-spaces, which emerge from the projection of $\Sigma^*$ into $Z$, are of the same class of mathematical object as Barnsley's fractal attractors (e.g. the widely reproduced fern). Using the method of (Grassberger & Procaccia, 1983), the correlation dimension of the attractor in Figure 2(g) was found to be about 1.4.

The link between work in complex dynamical systems and neural networks is well-established both on the neurobiological level (Skarda & Freeman, 1987) and on the mathematical level (Derrida & Meir, 1988; Huberman & Hogg, 1987; Kurten, 1987; Smolensky, 1986). This paper expands a theme from an earlier proposal to link them at the "cognitive" level (Pollack, 1989).

There is an interesting formal question, which has been brought out in the work of (Wolfram, 1984) and others on the universality of cellular automata, and more recently in the work of (Crutchfield & Young, 1989) on the descriptive complexity of bifurcating systems: What is the relationship between complex dynamics (of neural systems) and traditional measures of computational complexity? From this work and other supporting evidence, I venture the following hypothesis:

The state-space limit of a dynamical recognizer, as $\Sigma^* \rightarrow \Sigma^\infty$, is an Attractor, which is cut by a threshold (or similar decision) function. The complexity of the language recognized is regular if the cut falls between disjoint limit points or cycles, context-free if it cuts a "self-similar" (recursive) region, and context-sensitive if it cuts a "chaotic" (pseudo-random) region.

## Acknowledgements

This research has been partially supported by the Office of Naval Research under grant N00014-89-J-1200.

## Footnotes

[1] For the simple low dimensional dynamical systems usually studied, the "knob" or control parameter for such a bifurcation diagram is a scalar variable; here the control parameter is the entire 32-D vector of weights in the network, and back-propagation turns the knob!

[2] It can be argued that other FSA inducing methods get around this problem by presupposing rather than learning trap states.

[3] One reason I have succeeded in such low dimensional induction is because my architecture is a Mealy, rather than Moore Machine (Lee Giles, Personal Communication)

## References

Angluin, D. (1978). On the complexity of minimum inference of regular sets. *Information and Control, 39,* 337-350.

Barnsley, M. F., Ervin, V., Hardin, D. & Lancaster, J. (1985). Solution of an inverse problem for fractals and other sets. *Proceedings of the National Academy of Science, 83.*

Crutchfield, J. P & Young, K. (1989). Computation at the Onset of Chaos. In W. Zurek, (Ed.), *Complexity, Entropy and the Physics of INformation.* Reading, MA: Addison-Wesley.

Derrida, B. & Meir, R. (1988). Chaotic behavior of a layered neural network. *Phys. Rev. A, 38.*

Elman, J. L. (1990). Finding Structure in Time. *Cognitive Science, 14,* 179-212.

Giles, C. L., Sun, G. Z., Chen, H. H., Lee, Y. C. & Chen, D. (1990). Higher Order Recurrent Networks and Grammatical Inference. In D. Touretzky, (Ed.), *Advances in Neural Information Processing Systems.* Los Gatos, CA: Morgan Kaufman.

Grassberger, P. & Procaccia, I. (1983). Measuring the Strangeness of Strange Attractors. *Physica, 9D*, 189-208.

Huberman, B. A. & Hogg, T. (1987). Phase Transitions in Artificial Intelligence Systems. *Artificial Intelligence, 33*, 155-172.

Jordan, M. I. (1986). Serial Order: A Parallel Distributed Processing Approach. ICS report 8608, La Jolla: Institute for Cognitive Science, UCSD.

Kurten, K. E. (1987). Phase transitions in quasirandom neural networks. In *Institute of Electrical and Electronics Engineers First International Conference on Neural Networks*. San Diego, II-197-20.

McCulloch, W. S. & Pitts, W. (1943). A logical calculus of the ideas immanent in nervous activity. *Bulletin of Mathematical Biophysics, 5*, 115-133.

Pollack, J. B. (1987). Cascaded Back Propagation on Dynamic Connectionist Networks. In *Proceedings of the Ninth Conference of the Cognitive Science Society*. Seattle, 391-404.

Pollack, J. B. (1989). Implications of Recursive Distributed Representations. In D. Touretzky, (Ed.), *Advances in Neural Information Processing Systems*. Los Gatos, CA: Morgan Kaufman.

Pollack, J. B. (1990). Recursive Distributed Representation. *Artificial Intelligence, 46*, 77-105.

Pollack, J. B. (1990). The Induction of Dynamical Recognizers. Tech Report 90-JP-Automata, Columbus, OH 43210: LAIR, Ohio State University.

Rumelhart, D. E., Hinton, G. & Williams, R. (1986). Learning Internal Representations through Error Propagation. In D. E. Rumelhart, J. L. McClelland & the PDP research Group, (Eds.), *Parallel Distributed Processing: Experiments in the Microstructure of Cognition*, Vol. 1. Cambridge: MIT Press.

Servan-Schreiber, D., Cleeremans, A. & McClelland, J. L (1989). Encoding Sequential Structure in Simple Recurrent Networks. In D. Touretzky, (Ed.), *Advances in Neural Information Processing Systems*. Los Gatos, CA: Morgan Kaufman.

Skarda, C. A. & Freeman, W. J. (1987). How brains make chaos. *Brain & Behavioral Science, 10*.

Smolensky, P. (1986). Information Processing in Dynamical Systems: Foundations of Harmony Theory. In D. E. Rumelhart, J. L. McClelland & the PDP research Group, (Eds.), *Parallel Distributed Processing: Experiments in the Microstructure of Cognition*, Vol. 1. Cambridge: MIT Press.

Tomita, M. (1982). Dynamic construction of finite-state automata from examples using hill-climbing. In *Proceedings of the Fourth Annual Cognitive Science Conference*. Ann Arbor, MI, 105-108.

Wolfram, S. (1984). Universality and Complexity in Cellular Automata. *Physica, 10D*, 1-35.
